# Semidefinite Programming
# by Perceptron Learning

**Thore Graepel**    **Ralf Herbrich**
Microsoft Research Ltd., Cambridge, UK
*{thoreg,rherb}@microsoft.com*

**Andriy Kharechko**    **John Shawe-Taylor**
Royal Holloway, University of London, UK
*{ak03r,jst}@ecs.soton.ac.uk*

## Abstract

We present a modified version of the perceptron learning algorithm (PLA) which solves semidefinite programs (SDPs) in polynomial time. The algorithm is based on the following three observations: (i) Semidefinite programs are linear programs with infinitely many (linear) constraints; (ii) every linear program can be solved by a sequence of constraint satisfaction problems with linear constraints; (iii) in general, the perceptron learning algorithm solves a constraint satisfaction problem with linear constraints in finitely many updates. Combining the PLA with a probabilistic rescaling algorithm (which, on average, increases the size of the feasable region) results in a probabilistic algorithm for solving SDPs that runs in polynomial time. We present preliminary results which demonstrate that the algorithm works, but is not competitive with state-of-the-art interior point methods.

## 1   Introduction

Semidefinite programming (SDP) is one of the most active research areas in optimisation. Its appeal derives from important applications in combinatorial optimisation and control theory, from the recent development of efficient algorithms for solving SDP problems and the depth and elegance of the underlying optimisation theory [14], which covers linear, quadratic, and second-order cone programming as special cases.

Recently, semidefinite programming has been discovered as a useful toolkit in machine learning with applications ranging from pattern separation via ellipsoids [4] to kernel matrix optimisation [5] and transformation invariant learning [6].

Methods for solving SDPs have mostly been developed in an analogy to linear programming. Generalised *simplex-like algorithms* were developed for SDPs [11], but to the best of our knowledge are currently merely of theoretical interest. The *ellipsoid method* works by searching for a feasible point via repeatedly "halving" an ellipsoid that encloses the affine space of constraint matrices such that the centre of the ellipsoid is a feasible point [7]. However, this method shows poor performance in practice

as the running time usually attains its worst-case bound. A third set of methods for solving SDPs are *interior point methods* [14]. These methods minimise a linear function on convex sets provided the sets are endowed with self-concordant barrier functions. Since such a barrier function is known for SDPs, interior point methods are currently the most efficient method for solving SDPs in practice.

Considering the great generality of semidefinite programming and the complexity of state-of-the-art solution methods it is quite surprising that the forty year old simple perceptron learning algorithm [12] can be modified so as to solve SDPs. In this paper we present a combination of the perceptron learning algorithm (PLA) with a rescaling algorithm (originally developed for LPs [3]) that is able to solve semidefinite programs in polynomial time. We start with a short introduction into semidefinite programming and the perceptron learning algorithm in Section 2. In Section 3 we present our main algorithm together with some performance guarantees, whose proofs we only sketch due to space restrictions. While our numerical results presented in Section 4 are very preliminary, they do give insights into the workings of the algorithm and demonstrate that machine learning may have something to offer to the field of convex optimisation.

For the rest of the paper we denote matrices and vectors by bold face upper and lower case letters, e.g., $\mathbf{A}$ and $\mathbf{x}$. We shall use $\bar{\mathbf{x}} := \mathbf{x}/\|\mathbf{x}\|$ to denote the unit length vector in the direction of $\mathbf{x}$. The notation $\mathbf{A} \succeq \mathbf{0}$ is used to denote $\mathbf{x}'\mathbf{A}\mathbf{x} \geq 0$ for all $\mathbf{x}$, that is, $\mathbf{A}$ is positive semidefinite.

## 2 Learning and Convex Optimisation

### 2.1 Semidefinite Programming

In semidefinite programming a linear objective function is minimised over the image of an affine transformation of the cone of semidefinite matrices, expressed by linear matrix inequalities (LMI):

$$\underset{\mathbf{x} \in \mathbb{R}^n}{\text{minimise}} \quad \mathbf{c}'\mathbf{x} \quad \text{subject to} \quad \mathbf{F}(\mathbf{x}) := \mathbf{F}_0 + \sum_{i=1}^{n} x_i \mathbf{F}_i \succeq \mathbf{0}, \tag{1}$$

where $\mathbf{c} \in \mathbb{R}^n$ and $\mathbf{F}_i \in \mathbb{R}^{m \times m}$ for all $i \in \{0, \ldots, n\}$. The following proposition shows that semidefinite programs are a direct generalisation of linear programs.

**Proposition 1.** *Every semidefinite program is a linear program with infinitely many linear constraints.*

*Proof.* Obviously, the objective function in (1) is linear in $\mathbf{x}$. For any $\mathbf{u} \in \mathbb{R}^m$, define the vector $\mathbf{a_u} := (\mathbf{u}'\mathbf{F}_1\mathbf{u}, \ldots, \mathbf{u}'\mathbf{F}_n\mathbf{u})$. Then, the constraints in (1) can be written as

$$\forall \mathbf{u} \in \mathbb{R}^m : \quad \mathbf{u}'\mathbf{F}(\mathbf{x})\mathbf{u} \geq 0 \qquad \Leftrightarrow \qquad \forall \mathbf{u} \in \mathbb{R}^m : \quad \mathbf{x}'\mathbf{a_u} \geq -\mathbf{u}'\mathbf{F}_0\mathbf{u}. \tag{2}$$

This is a linear constraint in $\mathbf{x}$ for all $\mathbf{u} \in \mathbb{R}^m$ (of which there are infinitely many). $\square$

Since the objective function is linear in $\mathbf{x}$, we can solve an SDP by a sequence of semidefinite constraint satisfaction problems (CSPs) introducing the additional constraint $\mathbf{c}'\mathbf{x} \leq c_0$ and varying $c_0 \in \mathbb{R}$. Moreover, we have the following proposition.

**Proposition 2.** *Any SDP can be solved by a sequence of homogenised semidefinite CSPs of the following form:*

$$\text{find} \quad \mathbf{x} \in \mathbb{R}^{n+1} \quad \text{subject to} \quad \mathbf{G}(\mathbf{x}) := \sum_{i=0}^{n} x_i \mathbf{G}_i \succ \mathbf{0}.$$

---
**Algorithm 1** Perceptron Learning Algorithm
---
**Require:** A (possibly) infinite set $A$ of vectors $\mathbf{a} \in \mathbb{R}^n$
  Set $t \leftarrow 0$ and $\mathbf{x}_t = \mathbf{0}$
  **while** there exists $\mathbf{a} \in A$ such that $\mathbf{x}_t' \overline{\mathbf{a}} \leq 0$ **do**
    $\mathbf{x}_{t+1} = \mathbf{x}_t + \overline{\mathbf{a}}$
    $t \leftarrow t + 1$
  **end while**
  **return** $\mathbf{x}_t$
---

*Proof.* In order to make $\mathbf{F}_0$ and $c_0$ dependent on the optimisation variables, we introduce an auxiliary variable $x_0 > 0$; the solution to the original problem is given by $x_0^{-1} \cdot \mathbf{x}$. Moreover, we can repose the two linear constraints $c_0 x_0 - \mathbf{c}' \mathbf{x} \geq 0$ and $x_0 > 0$ as an LMI using the fact that a block-diagonal matrix is positive (semi)definite if and only if every block is positive (semi)definite. Thus, the following matrices are sufficient:

$$\mathbf{G}_0 = \begin{pmatrix} \mathbf{F}_0 & \mathbf{0} & \mathbf{0} \\ \mathbf{0}' & c_0 & 0 \\ \mathbf{0}' & 0 & 1 \end{pmatrix}, \qquad \mathbf{G}_i = \begin{pmatrix} \mathbf{F}_i & \mathbf{0} & \mathbf{0} \\ \mathbf{0} & -c_i & 0 \\ \mathbf{0} & 0 & 0 \end{pmatrix}.$$

Given an upper and a lower bound on the objective function, repeated bisection can be used to determine the solution in $\mathcal{O}(\log \frac{1}{\varepsilon})$ steps to accuracy $\varepsilon$. $\qquad \square$

In order to simplify notation, we will assume that $n \leftarrow n+1$ and $m \leftarrow m+2$ whenever we speak about a semidefinite CSP for an SDP in $n$ variables with $\mathbf{F}_i \in \mathbb{R}^{m \times m}$.

## 2.2 Perceptron Learning Algorithm

The perceptron learning algorithm (PLA) [12] is an online procedure which finds a linear separation of a set of points from the origin (see Algorithm 1). In machine learning this algorithm is usually applied to two sets $A_{+1}$ and $A_{-1}$ of points labelled $+1$ and $-1$ by multiplying every data vector $\mathbf{a}_i$ by its class label[1]; the resulting vector $\mathbf{x}_t$ (often referred to as the weight vector in perceptron learning) is then read as the normal of a hyperplane which separates the sets $A_{+1}$ and $A_{-1}$.

A remarkable property of the perceptron learning algorithm is that the total number $t$ of updates is independent of the cardinality of $A$ but can be upper bounded simply in terms of the following quantity

$$\rho(A) := \max_{\mathbf{x} \in \mathbb{R}^n} \rho(A, \mathbf{x}) := \max_{\mathbf{x} \in \mathbb{R}^n} \min_{\mathbf{a} \in A} \overline{\mathbf{a}}' \overline{\mathbf{x}}.$$

This quantity is known as the (normalised) *margin* of $A$ in the machine learning community or as the *radius* of the feasible region in the optimisation community. It quantifies the radius of the largest ball that can be fitted in the convex region enclosed by all $\mathbf{a} \in A$ (the so-called *feasible set*). Then, the perceptron convergence theorem [10] states that $t \leq \rho^{-2}(A)$.

For the purpose of this paper we observe that Algorithm 1 solves a linear CSP where the linear constraints are given by the vectors $\mathbf{a} \in A$. Moreover, by the last argument we have the following proposition.

**Proposition 3.** *If the feasible set has a positive radius, then the perceptron learning algorithm solves a linear CSP in finitely many steps.*

It is worth mentioning that in the last few decades a series of modified PLAs $\mathcal{A}$ have been developed (see [2] for a good overview) which mainly aim at guaranteeing

**Algorithm 2** Rescaling algorithm

---

**Require:** A maximal number $T \in \mathbb{N}^+$ of steps and a parameter $\sigma \in \mathbb{R}^+$

   Set $\mathbf{y}$ uniformly at random in $\{\mathbf{z} \; : \; \|\mathbf{z}\| = 1\}$

   **for** $t = 0, \ldots, T$ **do**

      Find $\mathbf{a_u}$ such that $\bar{\mathbf{y}}'\bar{\mathbf{a}}_\mathbf{u} := \frac{\mathbf{u}'\mathbf{G}(\bar{\mathbf{y}})\mathbf{u}}{\sqrt{\sum_{j=1}^{n}(\mathbf{u}'\mathbf{G}_j\mathbf{u})^2}} \leq -\sigma$ ($\mathbf{u} \approx$ smallest EV of $\mathbf{G}(\bar{\mathbf{y}})$)

      **if** $\mathbf{u}$ does not exists **then**

         Set $\forall i \in \{1, \ldots, n\} : \; \mathbf{G}_i \leftarrow \mathbf{G}_i + \bar{y}_i \mathbf{G}(\bar{\mathbf{y}});$ **return y**

      **end if**

      $\mathbf{y} \leftarrow \mathbf{y} - (\mathbf{y}'\bar{\mathbf{a}}_\mathbf{u})\,\bar{\mathbf{a}}_\mathbf{u}; \; t \leftarrow t + 1$

   **end for**

   **return** *unsolved*

---

not only feasibility of the solution $\mathbf{x}_t$ but also a lower bound on $\rho(A, \mathbf{x}_t)$. These guarantees usually come at the price of a slightly larger mistake bound which we shall denote by $M(\mathcal{A}, \rho(A))$, that is, $t \leq M(\mathcal{A}, \rho(A))$.

## 3 Semidefinite Programming by Perceptron Learning

If we combine Propositions 1, 2 and 3 together with Equation (2) we obtain a perceptron algorithm that sequentially solves SDPs. However, there remain two problems:

1. How do we find a vector $\mathbf{a} \in A$ such that $\mathbf{x}'\mathbf{a} \leq 0$?

2. How can we make the running time of this algorithm polynomial in the description length of the data?[2]

In order to address the first problem we notice that $A$ in Algorithm 1 is not explicitly given but is defined by virtue of

$$A(\mathbf{G}_1, \ldots, \mathbf{G}_n) := \{\mathbf{a_u} := (\mathbf{u}'\mathbf{G}_1\mathbf{u}, \ldots, \mathbf{u}'\mathbf{G}_n\mathbf{u}) \mid \mathbf{u} \in \mathbb{R}^m\} \,.$$

Hence, finding a vector $\mathbf{a_u} \in A$ such that $\mathbf{x}'\mathbf{a_u} \leq 0$ is equivalent to identifying a vector $\mathbf{u} \in \mathbb{R}^m$ such that

$$\sum_{i=1}^{n} x_i \mathbf{u}'\mathbf{G}_i\mathbf{u} = \mathbf{u}'\mathbf{G}(\mathbf{x})\,\mathbf{u} \leq 0 \,.$$

One possible way of finding such a vector $\mathbf{u}$ (and consequently $\mathbf{a_u}$) for the current solution $\mathbf{x}_t$ in Algorithm 1 is to calculate the eigenvector corresponding to the smallest eigenvalue of $\mathbf{G}(\mathbf{x}_t)$; if this eigenvalue is positive, the algorithm stops and outputs $\mathbf{x}_t$. Note, however, that computationally easier procedures can be applied to find a suitable $\mathbf{u} \in \mathbb{R}^m$ (see also Section 4).

The second problem requires us to improve the dependency of the runtime from $\mathcal{O}(\rho^{-2})$ to $\mathcal{O}(-\log(\rho))$. To this end we employ a probabilistic *rescaling* algorithm (see Algorithm 2) which was originally developed for LPs [3]. The purpose of this algorithm is to enlarge the feasible region (in terms of $\rho(A(\mathbf{G}_1, \ldots, \mathbf{G}_n))$) by a constant factor $\kappa$, on average, which would imply a decrease in the number of updates of the perceptron algorithm exponential in the number of calls to this rescaling algorithm. This is achieved by running Algorithm 2. If the algorithm does not return *unsolved* the rescaling procedure on the $\mathbf{G}_i$ has the effect that $\mathbf{a_u}$ changes into $\mathbf{a_u} + (\bar{\mathbf{y}}'\mathbf{a_u})\,\bar{\mathbf{y}}$ for every $\mathbf{u} \in \mathbb{R}^m$. In order to be able to reconstruct the solution $\mathbf{x}_t$ to the original problem, whenever we rescale the $\mathbf{G}_i$ we need to remember the vector $\mathbf{y}$ used for rescaling. In Figure 1 we have shown the effect of rescaling for three linear con-

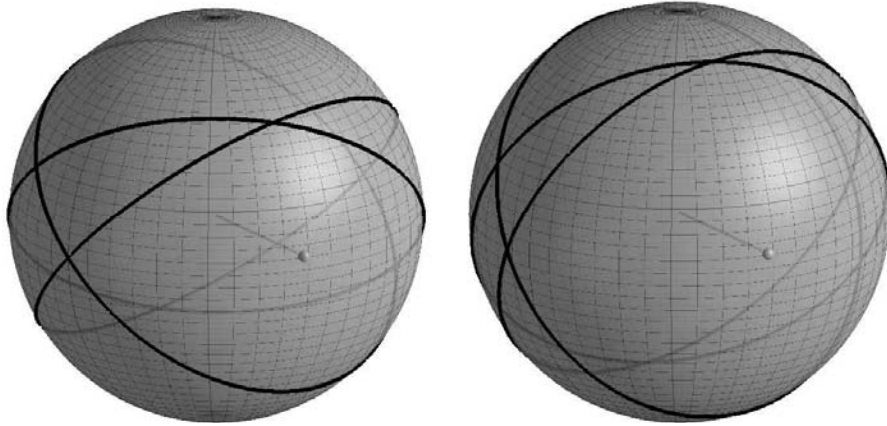

Figure 1: Illustration of the rescaling procedure. Shown is the feasible region and one feasible point before (**left**) and after (**left**) rescaling with the feasible point.

straints in $\mathbb{R}^3$. The main idea of Algorithm 2 is to find a vector $\mathbf{y}$ that is $\sigma$-close to the current feasible region and hence leads to an increase in its radius when used for rescaling. The following property holds for Algorithm 2.

**Theorem 1.** *Assume Algorithm 2 did not return* unsolved. *Let* $\sigma \leq \frac{1}{32n}$, $\rho$ *be the radius of the feasible set before rescaling and* $\rho'$ *be the radius of the feasible set after rescaling and assume that* $\rho \leq \frac{1}{4n}$. *Then*

1. *$\rho' \geq \left(1 - \frac{1}{16n}\right) \rho$ with probability at most $\frac{3}{4}$.*

2. *$\rho' \geq \left(1 + \frac{1}{4n}\right) \rho$ with probability at least $\frac{1}{4}$.*

The probabilistic nature of the theorem stems from the fact that the rescaling can only be shown to increase the size of the feasible region if the (random) initial value $\mathbf{y}$ already points sufficiently closely to the feasible region. A consequence of this theorem is that, on average, the radius increases by $\kappa = (1 + 1/64n) > 1$. Algorithm 3 combines rescaling and perceptron learning, which results in a probabilistic polynomial runtime algorithm[3] which alternates between calls to Algorithm 1 and 2 . This algorithm may return *infeasible* in two cases: either $T_i$ many calls to Algorithm 2 have returned *unsolved* or $L$ many calls of Algorithm 1 together with rescaling have not returned a solution. Each of these two conditions can either happen because of an "unlucky" draw of $\mathbf{y}$ in Algorithm 2 or because $\rho\left(A\left(\mathbf{G}_1, \ldots, \mathbf{G}_n\right)\right)$ is too small. Following the argument in [3] one can show that for $L = -2048n \cdot \ln\left(\rho_{\min}\right)$ the total probability of returning *infeasible* despite $\rho\left(A\left(\mathbf{G}_1, \ldots, \mathbf{G}_n\right)\right) > \rho_{\min}$ cannot exceed $\exp\left(-n\right)$.

## 4    Experimental Results

The experiments reported in this section fall into two parts. Our initial aim was to demonstrate that the method works in practice and to assess its efficacy on a

**Algorithm 3** Positive Definite Perceptron Algorithm

---

**Require:** $\mathbf{G}_1, \ldots, \mathbf{G}_n \in \mathbb{R}^{m \times m}$ and maximal number of iteration $L \in \mathbb{N}^+$

  Set $\mathbf{B} = \mathbf{I}_n$

  **for** $i = 1, \ldots, L$ **do**

    Call Algorithm 1 for at most $M\left(\mathcal{A}, \frac{1}{4n}\right)$ many updates

    **if** Algorithm 1 converged **then return Bx**

    Set $\delta_i = \frac{3}{\pi^2 i^2}$ and $T_i = \frac{\ln(\delta_i)}{\ln\left(\frac{3}{4}\right)}$

    **for** $j = 1, \ldots, T_i$ **do**

      Call Algorithm 2 with $T = 1024n^2 \ln(n)$ and $\sigma = \frac{1}{32n}$

      **if** Algorithm 2 returns $\mathbf{y}$ **then** $\mathbf{B} \leftarrow \mathbf{B}\left(\mathbf{I}_n + \overline{\mathbf{y}}\mathbf{y}'\right)$; goto the outer for-loop

    **end for**

    **return** *infeasible*

  **end for**

  **return** *infeasible*

---

benchmark example from graph bisection [1].

These experiments would also indicate how competitive the baseline method is when compared to other solvers. The algorithm was implemented in MATLAB and all of the experiments were run on 1.7GHz machines. The time taken can be compared with a standard method SDPT3 [13] partially implemented in C but running under MATLAB.

We considered benchmark problems arising from semidefinite relaxations to the MAXCUT problems of weighted graphs, which is posed as finding a maximum weight bisection of a graph. The benchmark MAXCUT problems have the following relaxed SDP form (see [8]):

$$\underset{\mathbf{x} \in \mathbb{R}^n}{\text{minimise}} \quad \mathbf{1}'\mathbf{x} \quad \text{subject to} \quad \underbrace{-\frac{1}{4}\left(\text{diag}(\mathbf{C}\mathbf{1}) - \mathbf{C}\right)}_{\mathbf{F}_0} + \underbrace{\text{diag}\left(\mathbf{x}\right)}_{\sum_i x_i \mathbf{F}_i} \succeq \mathbf{0}, \qquad (3)$$

where $\mathbf{C} \in \mathbb{R}^{n \times n}$ is the adjacency matrix of the graph with $n$ vertices.

The benchmark used was 'mcp100' provided by SDPLIB 1.2 [1]. For this problem, $n = 100$ and it is known that the optimal value of the objective function equals 226.1574. The baseline method used the bisection approach to identify the critical value of the objective, referred to throughout this section as $c_0$.

Figure 2 (left) shows a plot of the time per iteration against the value of $c_0$ for the first four iterations of the bisection method. As can be seen from the plots the time taken by the algorithm for each iteration is quite long, with the time of the fourth iteration being around 19,000 seconds. The initial value of 999 for $c_0$ was found without an objective constraint and converged within 0.012 secs. The bisection then started with the lower (infeasible) value of 0 and the upper value of 999. Iteration 1 was run with $c_0 = 499.5$, but the feasible solution had an objective value of 492. This was found in just 617 secs. The second iteration used a value of $c_0 = 246$ slightly above the optimum of 226. The third iteration was infeasible but since it was quite far from the optimum, the algorithm was able to deduce this fact quite quickly. The final iteration was also infeasible, but much closer to the optimal value. The running time suffered correspondingly taking 5.36 hours. If we were to continue the next iteration would also be infeasible but closer to the optimum and so would take even longer.

The first experiment demonstrated several things. First, that the method does indeed work as predicted; secondly, that the running times are very far from being

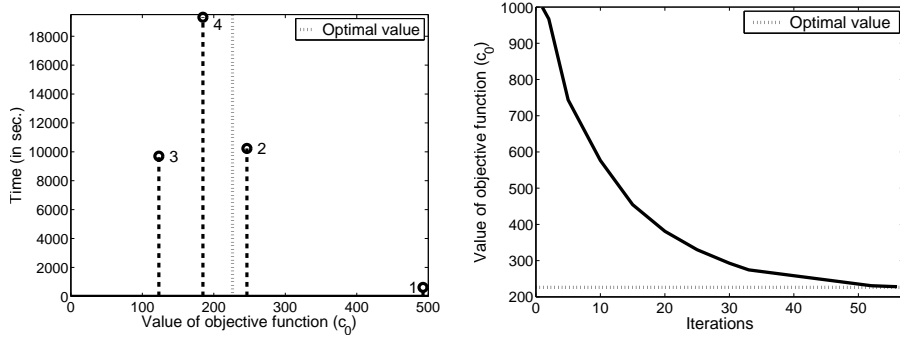

Figure 2: (**Left**) Four iterations of the bisection method showing time taken per iteration (outer for loop in Algorithm 3) against the value of the objective constraint. (**Right**) Decay of the attained objective function value while iterating through Algorithm 3 with a non-zero threshold of $\tau = 500$.

competitive (SDPT3 takes under 12 seconds to solve this problem) and thirdly that the running times increase as the value of $c_0$ approaches the optimum with those iterations that must prove infeasibility being more costly than those that find a solution.

The final observation prompted our first adaptation of the base algorithm. Rather than perform the search using the bisection method we implemented a non-zero threshold on the objective constraint (see the while-statement in Algorithm 1). The value of this threshold is denoted $\tau$, following the notation introduced in [9].

Using a value of $\tau = 500$ ensured that when a feasible solution is found, its objective value is significantly below that of the objective constraint $c_0$. Figure 2 (right) shows the values of $c_0$ as a function of the outer for-loops (iterations); the algorithm eventually approached its estimate of the optimal value at 228.106. This is within 1% of the optimum, though of course iterations could have been continued. Despite the clear convergence, using this approach the running time to an accurate estimate of the solution is still prohibitive because overall the algorithm took approximately 60 hours of CPU time to find its solution.

A profile of the execution, however, revealed that up to 93% of the execution time is spent in the eigenvalue decomposition to identify $\mathbf{u}$. Observe that we do not need a minimal eigenvector to perform an update, simply a vector $\mathbf{u}$ satisfying

$$\mathbf{u}'\mathbf{G}(\mathbf{x})\mathbf{u} < 0 \qquad (4)$$

Cholesky decomposition will either return $\mathbf{u}$ satisfying (4) or it will converge indicating that $\mathbf{G}(\mathbf{x})$ is psd and Algorithm 1 has converged.

## 5  Conclusions

Semidefinite programming has interesting applications in machine learning. In turn, we have shown how a simple learning algorithm can be modified to solve higher order convex optimisation problems such as semidefinite programs. Although the experimental results given here suggest the approach is far from computationally competitive, the insights gained may lead to effective algorithms in concrete applications in the same way that for example SMO is a competitive algorithm for solving quadratic programming problems arising from support vector machines. While the

optimisation setting leads to the somewhat artificial and inefficient bisection method the positive definite perceptron algorithm excels at solving positive definite CSPs as found, e.g., in problems of transformation invariant pattern recognition as solved by Semidefinite Programming Machines [6]. In future work it will be of interest to consider the combined primal-dual problem at a predefined level $\varepsilon$ of granularity so as to avoid the necessity of bisection search.

**Acknowledgments** We would like to thank J. Kandola, J. Dunagan, and A. Ambroladze for interesting discussions. This work was supported by EPSRC under grant number GR/R55948 and by Microsoft Research Cambridge.

## Footnotes

[1]Note that sometimes the update equation is given using the unnormalised vector $\mathbf{a}$.

[2]Note that polynomial runtime is only guaranteed if $\rho^{-2}(A(\mathbf{G}_1, \ldots, \mathbf{G}_n))$ is bounded by a polynomial function of the description length of the data.

[3]Note that we assume that the optimisation problem in line 3 of Algorithm 2 can be solved in polynomial time with algorithms such as Newton-Raphson.

## References

[1] B. Borchers. SDPLIB 1.2, A library of semidefinite programming test problems. *Optimization Methods and Software*, 11(1):683–690, 1999.

[2] R. O. Duda, P. E. Hart, and D. G. Stork. *Pattern Classification and Scene Analysis*. John Wiley and Sons, New York, 2001. Second edition.

[3] J. Dunagan and S. Vempala. A polynomial-time rescaling algorithm for solving linear programs. Technical Report MSR-TR-02-92, Microsoft Research, 2002.

[4] F. Glineur. Pattern separation via ellipsoids and conic programming. Mémoire de D.E.A., Faculté Polytechnique de Mons, Mons, Belgium, Sept. 1998.

[5] T. Graepel. Kernel matrix completion by semidefinite programming. In J. R. Dorronsoro, editor, *Proceedings of the International Conference on Neural Networks, ICANN2002*, Lecture Notes in Computer Science, pages 694–699. Springer, 2002.

[6] T. Graepel and R. Herbrich. Invariant pattern recognition by Semidefinite Programming Machines. In S. Thrun, L. Saul, and B. Schölkopf, editors, *Advances in Neural Information Processing Systems 16*. MIT Press, 2004.

[7] M. Grötschel, L. Lovász, and A. Schrijver. *Geometric Algorithms and Combinatorial Optimization*, volume 2 of *Algorithms and Combinatorics*. Springer-Verlag, 1988.

[8] C. Helmberg. Semidefinite programming for combinatorial optimization. Technical Report ZR-00-34, Konrad-Zuse-Zentrum für Informationstechnik Berlin, Oct. 2000.

[9] Y. Li, H. Zaragoza, R. Herbrich, J. Shawe-Taylor, and J. Kandola. The perceptron algorithm with uneven margins. In *Proceedings of the International Conference of Machine Learning (ICML'2002)*, pages 379–386, 2002.

[10] A. B. J. Novikoff. On convergence proofs on perceptrons. In *Proceedings of the Symposium on the Mathematical Theory of Automata*, volume 12, pages 615–622. Polytechnic Institute of Brooklyn, 1962.

[11] G. Pataki. Cone-LP's and semi-definite programs: facial structure, basic solutions, and the simplex method. Technical Report GSIA, Carnegie Mellon University, 1995.

[12] F. Rosenblatt. The perceptron: A probabilistic model for information storage and organization in the brain. *Psychological Review*, 65(6):386–408, 1958.

[13] K. C. Toh, M. Todd, and R. Tütüncü. SDPT3 – a MATLAB software package for semidefinite programming. Technical Report TR1177, Cornell University, 1996.

[14] L. Vandenberghe and S. Boyd. Semidefinite programming. *SIAM Review*, 38(1):49–95, 1996.
